# Goal-directed decision making in prefrontal cortex: A computational framework

**Matthew Botvinick**
Princeton Neuroscience Institute and
Department of Psychology, Princeton University
Princeton, NJ 08540
*matthewb@princeton.edu*

**James An**
Computer Science Department
Princeton University
Princeton, NJ 08540
*an@princeton.edu*

## Abstract

Research in animal learning and behavioral neuroscience has distinguished between two forms of action control: a habit-based form, which relies on stored action values, and a goal-directed form, which forecasts and compares action outcomes based on a model of the environment. While habit-based control has been the subject of extensive computational research, the computational principles underlying goal-directed control in animals have so far received less attention. In the present paper, we advance a computational framework for goal-directed control in animals and humans. We take three empirically motivated points as founding premises: (1) Neurons in dorsolateral prefrontal cortex represent action policies, (2) Neurons in orbitofrontal cortex represent rewards, and (3) Neural computation, across domains, can be appropriately understood as performing structured probabilistic inference. On a purely computational level, the resulting account relates closely to previous work using Bayesian inference to solve Markov decision problems, but extends this work by introducing a new algorithm, which provably converges on optimal plans. On a cognitive and neuroscientific level, the theory provides a unifying framework for several different forms of goal-directed action selection, placing emphasis on a novel form, within which orbitofrontal reward representations directly drive policy selection.

## 1    Goal-directed action control

In the study of human and animal behavior, it is a long-standing idea that reward-based decision making may rely on two qualitatively different mechanisms. In *habit*-based decision making, stimuli elicit reflex-like responses, shaped by past reinforcement [1]. In *goal-directed* or *purposive* decision making, on the other hand, actions are selected based on a prospective consideration of possible outcomes and future lines of action [2]. Over the past twenty years or so, the attention of cognitive neuroscientists and computationally minded psychologists has tended to focus on habit-based control, due in large part to interest in potential links between dopaminergic function and temporal-difference algorithms for reinforcement learning. However, a resurgence of interest in purposive action selection is now being driven by innovations in animal behavior research, which have yielded powerful new behavioral assays [3], and revealed specific effects of focal neural damage on goal-directed behavior [4].

In discussing some of the relevant data, Daw, Niv and Dayan [5] recently pointed out the close relationship between purposive decision making, as understood in the behavioral sciences, and *model-based* methods for the solution of Markov decision problems (MDPs), where action policies are derived from a joint analysis of a transition function (a mapping

from states and actions to outcomes) and a reward function (a mapping from states to rewards). Beyond this important insight, little work has yet been done to characterize the computations underlying goal-directed action selection (though see [6, 7]). As discussed below, a great deal of evidence indicates that purposive action selection depends critically on a particular region of the brain, the prefrontal cortex. However, it is currently a critical, and quite open, question what the relevant computations within this part of the brain might be.

Of course, the basic computational problem of formulating an optimal policy given a model of an MDP has been extensively studied, and there is no shortage of algorithms one might consider as potentially relevant to prefrontal function (e.g., value iteration, policy iteration, backward induction, linear programming, and others). However, from a cognitive and neuroscientific perspective, there is one approach to solving MDPs that it seems particularly appealing to consider. In particular, several researchers have suggested methods for solving MDPs through *probabilistic inference* [8-12]. The interest of this idea, in the present context, derives from a recent movement toward framing human and animal information processing, as well as the underlying neural computations, in terms of structured probabilistic inference [13, 14]. Given this perspective, it is inviting to consider whether goal-directed action selection, and the neural mechanisms that underlie it, might be understood in those same terms.

One challenge in investigating this possibility is that previous research furnishes no 'off-the-shelf' algorithm for solving MDPs through probabilistic inference that both provably yields optimal policies and aligns with what is known about action selection in the brain. We endeavor here to start filling in that gap. In the following section, we introduce an account of how goal-directed action selection can be performed based on probabilisitic inference, within a network whose components map grossly onto specific brain structures. As part of this account, we introduce a new algorithm for solving MDPs through Bayesian inference, along with a convergence proof. We then present results from a set of simulations illustrating how the framework would account for a variety of behavioral phenomena that are thought to involve purposive action selection.

## 2      Computational model

As noted earlier, the prefrontal cortex (PFC) is believed to play a pivotal role in purposive behavior. This is indicated by a broad association between prefrontal lesions and impairments in goal-directed action in both humans (see [15]) and animals [4]. Single-unit recording and other data suggest that different sectors of PFC make distinct contributions. In particular, neurons in dorsolateral prefrontal cortex (DLPFC) appear to encode task-specific mappings from stimuli to responses (e.g., [16]): "task representations," in the language of psychology, or "policies" in the language of dynamic programming. Although there is some understanding of how policy representations in DLPFC may guide action execution [15], little is yet known about how these representations are themselves selected. Our most basic proposal is that DLPFC policy representations are selected in a prospective, model-based fashion, leveraging information about action-outcome contingencies (i.e., the transition function) and about the incentive value associated with specific outcomes or states (the reward function). There is extensive evidence to suggest that state-reward associations are represented in another area of the PFC, the orbitofrontal cortex (OFC) [17, 18]. As for the transition function, although it is clear that the brain contains detailed representations of action-outcome associations [19], their anatomical localization is not yet entirely clear. However, some evidence suggests that the enviromental effects of simple actions may be represented in inferior fronto-parietal cortex [20], and there is also evidence suggesting that medial temporal structures may be important in forecasting action outcomes [21].

As detailed in the next section, our model assumes that policy representations in DLPFC, reward representations in OFC, and representations of states and actions in other brain regions, are coordinated within a network structure that represents their causal or statistical interdependencies, and that policy selection occurs, within this network, through a process of probabilistic inference.

### 2.1    Architecture

The implementation takes the form of a directed graphical model [22], with the layout shown in Figure 1. Each node represents a discrete random variable. *State* variables ($s$),

representing the set of *m* possible world states, serve the role played by parietal and medial temporal cortices in representing action outcomes. *Action* variables (*a*) representing the set of available actions, play the role of high-level cortical motor areas involved in the programming of action sequences. *Policy* variables (*π*), each repre-senting the set of all deterministic policies associated with a specific state, capture the representational role of DLPFC. Local and global *utility* variables, described further below, capture the role of OFC in representing incentive value. A separate set of nodes is included for each discrete time-step up to the planning horizon.

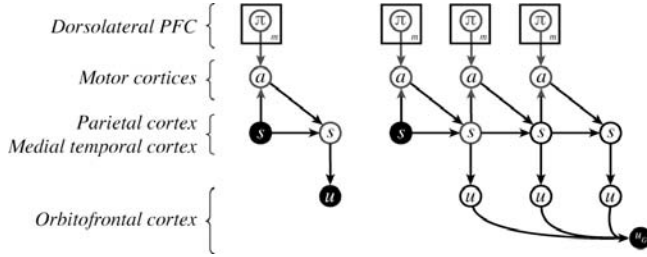

Fig 1. Left: Single-step decision. Right: Sequential decision. Each time-slice includes a set of *m* policy nodes.

The conditional probabilities associated with each variable are represented in tabular form. State probabilities are based on the state and action variables in the preceding time-step, and thus encode the transition function. Action probabilities depend on the current state and its associated policy variable. Utilities depend only on the current state. Rather than representing reward magnitude as a continuous variable, we adopt an approach introduced by [23], representing reward through the posterior probability of a binary variable (*u*). States associated with large positive reward raise $p(u)$ (i.e, $p(u=1|s)$) near to one; states associated with large negative rewards reduce $p(u)$ to near zero. In the simulations reported below, we used a simple linear transformation to map from scalar reward values to $p(u)$:

$$p(u|s_i) = \frac{1}{2}\left(\frac{R(s_i)}{r_{max}} + 1\right), \quad r_{max} \equiv \max_j |R(s_j)| \tag{1}$$

In situations involving sequential actions, expected returns from different time-steps must be integrated into a global representation of expected value. In order to accomplish this, we employ a technique proposed by [8], introducing a "global" utility variable ($u_G$). Like *u*, this is a binary random variable, but associated with a posterior probability determined as:[1]

$$p(u_G) = \frac{1}{N}\sum_i p(u_i) \tag{2}$$

where *N* is the number of *u* nodes. The network as whole embodies a generative model for instrumental action. The basic idea is to use this model as a substrate for probabilistic inference, in order to arrive at optimal policies. There are three general methods for accomplishing this, which correspond three forms of *query*. First, a desired outcome state can be identified, by treating one of the state variables (as well as the initial state variable) as observed (see [9] for an application of this approach). Second, the expected return for specific plans can be evaluated and compared by conditioning on specific sets of values over the policy nodes (see [5, 21]). However, our focus here is on a less obvious possibility, which is to condition directly on the utility variable $u_G$, as explained next.

## 2.2    Policy selection by probabilistic inference: an iterative algorithm

Cooper [23] introduced the idea of inferring optimal decisions in influence diagrams by treating utility nodes into binary random variables and then conditioning on these variables. Although this technique has been adopted in some more recent work [9, 12], we are aware of no application that guarantees optimal decisions, in the expected-reward sense, in multi-step tasks. We introduce here a simple algorithm that does furnish such a guarantee. The procedure is as follows: (1) Initialize the policy nodes with any set of non-deterministic priors. (2) Treating the initial state and $u_G$ as observed variables ($u_G = 1$),[2] use standard belief

propagation (or a comparable algorithm) to infer the posterior distributions over all policy nodes. (3) Set the *prior* distributions over the policy nodes to the values (posteriors) obtained in step 2. (4) Go to step 2. The next two sections present proofs of monotonicity and convergence for this algorithm.

## 2.2.1  Monotonicity

We show first that, at each policy node, the probability associated with the optimal policy will rise on every iteration. Define $\pi^*$ as follows:

$$p\left(u_G|\pi^*,\pi^+\right) > p\left(u_G|\pi',\pi^+\right), \forall \pi' \neq \pi^* \tag{3}$$

where $\pi^+$ is the current set of probability distributions at all policy nodes on subsequent time-steps. (Note that we assume here, for simplicity, that there is a unique optimal policy.) The objective is to establish that:

$$p\left(\pi_t^*\right) > p\left(\pi_{t-1}^*\right) \tag{4}$$

where $t$ indexes processing iterations. The dynamics of the network entail that

$$p\left(\pi_t\right) = p\left(\pi_{t-1}|u_G\right) \tag{5}$$

where $\pi$ represents any value (i.e., policy) of the decision node being considered. Substituting this into (4) gives

$$p\left(\pi_{t-1}^*|u_G\right) > p\left(\pi_{t-1}^*\right) \tag{6}$$

From this point on the focus is on a single iteration, which permits us to omit the relevant subscripts. Applying Bayes' law to (6) yields

$$\frac{p\left(u_G|\pi^*\right)p\left(\pi^*\right)}{\sum_{\pi} p\left(u_G|\pi\right)p\left(\pi\right)} > p\left(\pi^*\right) \tag{7}$$

Canceling, and bringing the denominator up, this becomes

$$p\left(u_G|\pi^*\right) > \sum_{\pi} p\left(u_G|\pi\right)p\left(\pi\right) \tag{8}$$

Rewriting the left hand side, we obtain

$$\sum_{\pi} p\left(u_G|\pi^*\right)p\left(\pi\right) > \sum_{\pi} p\left(u_G|\pi\right)p\left(\pi\right) \tag{9}$$

Subtracting and further rearranging:

$$\sum_{\pi}\left[p\left(u_G|\pi^*\right) - p\left(u_G|\pi\right)\right]p\left(\pi\right) > 0 \tag{10}$$

$$\left[p\left(u_G|\pi^*\right) - p\left(u_G|\pi^*\right)\right]p\left(\pi^*\right) + \sum_{\pi' \neq \pi^*}\left[p\left(u_G|\pi^*\right) - p\left(u_G|\pi'\right)\right]p\left(\pi'\right) > 0 \tag{11}$$

$$\sum_{\pi' \neq \pi^*}\left[p\left(u_G|\pi^*\right) - p\left(u_G|\pi'\right)\right]p\left(\pi'\right) > 0 \tag{12}$$

Note that this last inequality (12) follows from the definition of $\pi^*$.

*Remark:* Of course, the identity of $\pi^*$ depends on $\pi^+$. In particular, the policy $\pi^*$ will only be part of a globally optimal plan if the set of choices $\pi^+$ is optimal. Fortunately, this requirement is guaranteed to be met, as long as no upper bound is placed on the number of processing cycles. Recalling that we are considering only finite-horizon problems, note that for policies leading to states with no successors, $\pi^+$ is empty. Thus $\pi^*$ at the relevant policy nodes is fixed, and is guaranteed to be part of the optimal policy. The proof above shows that $\pi^*$ will continuously rise. Once it reaches a maximum, $\pi^*$ at immediately preceding decisions will perforce fit with the globally optimal policy. The process works backward, in the fashion of backward induction.

## 2.2.2 Convergence

Continuing with the same notation, we show now that

$$\lim_{t\to\infty} p_t\left(\pi^* | u_G\right) = 1 \tag{13}$$

Note that, if we apply Bayes' law recursively,

$$p_t\left(\pi^* | u_G\right) = \frac{p\left(u_G | \pi^*\right) p_t\left(\pi^*\right)}{p_i\left(u_G\right)} = \frac{p\left(u_G | \pi^*\right)^2 p_{t-1}\left(\pi^*\right)}{p_i\left(u_G\right) p_{t-1}\left(u_G\right)} = \frac{p\left(u_G | \pi^*\right)^3 p_{t-2}\left(\pi^*\right)}{p_t\left(u_G\right) p_{t-1}\left(u_G\right) p_{t-2}\left(u_G\right)} \cdots \tag{14}$$

Thus,

$$p_1\left(\pi^* | u_G\right) = \frac{p\left(u_G | \pi^*\right) p_1\left(\pi^*\right)}{p_1\left(u_G\right)}, \quad p_2\left(\pi^* | u_G\right) = \frac{p\left(u_G | \pi^*\right)^2 p_1\left(\pi^*\right)}{p_2\left(u_G\right) p_1\left(u_G\right)}, \quad p_3\left(\pi^* | u_G\right) = \frac{p\left(u_G | \pi^*\right)^3 p_1\left(\pi^*\right)}{p_3\left(u_G\right) p_2\left(u_G\right) p_1\left(u_G\right)}, \tag{15}$$

and so forth. Thus, what we wish to prove is

$$\frac{p\left(u_G | \pi^*\right)^\infty p_1\left(\pi^*\right)}{\prod\limits_{t=1}^{\infty} p_t\left(u_G\right)} = 1 \tag{16}$$

or, rearranging,

$$\prod_{t=1}^{\infty} \frac{p_t\left(u_G\right)}{p\left(u_G | \pi^*\right)} = p_1\left(\pi^*\right). \tag{17}$$

Note that, given the stipulated relationship between $p(\pi)$ on each processing iteration and $p(\pi | u_G)$ on the previous iteration,

$$p_t\left(u_G\right) = \sum_\pi p\left(u_G | \pi\right) p_t\left(\pi\right) = \sum_\pi p\left(u_G | \pi\right) p_{t-1}\left(\pi | u_G\right) = \frac{\sum\limits_\pi p\left(u_G | \pi\right)^2 p_{t-1}\left(\pi\right)}{p_{t-1}\left(u_G\right)}$$

$$= \frac{\sum\limits_\pi p\left(u_G | \pi\right)^3 p_{t-1}\left(\pi\right)}{p_{t-1}\left(u_G\right) p_{t-2}\left(u_G\right)} = \frac{\sum\limits_\pi p\left(u_G | \pi\right)^4 p_{t-1}\left(\pi\right)}{p_{t-1}\left(u_G\right) p_{t-2}\left(u_G\right) p_{t-3}\left(u_G\right)} \cdots \tag{18}$$

With this in mind, we can rewrite the left hand side product in (17) as follows:

$$\frac{p_1\left(u_G\right)}{p\left(u_G | \pi^*\right)} \cdot \frac{\sum\limits_\pi p\left(u_G | \pi\right)^2 p_1\left(\pi\right)}{p\left(u_G | \pi^*\right) p_1\left(u_G\right)} \cdot \frac{\sum\limits_\pi p\left(u_G | \pi\right)^3 p_1\left(\pi\right)}{p\left(u_G | \pi^*\right) p_1\left(u_G\right) p_2\left(u_G\right)} \cdot \frac{\sum\limits_\pi p\left(u_G | \pi\right)^4 p_1\left(\pi\right)}{p\left(u_G | \pi^*\right) p_1\left(u_G\right) p_2\left(u_G\right) p_3\left(u_G\right)} \cdots \tag{19}$$

Note that, given (18), the numerator in each factor of (19) cancels with the denominator in the subsequent factor, leaving only $p(u_G|\pi^*)$ in that denominator. The expression can thus be rewritten as

$$\frac{1}{p\left(u_G | \pi^*\right)} \cdot \frac{1}{p\left(u_G | \pi^*\right)} \cdot \frac{1}{p\left(u_G | \pi^*\right)} \cdot \frac{\sum\limits_\pi p\left(u_G | \pi\right)^4 p_1\left(\pi\right)}{p\left(u_G | \pi^*\right)} \cdots = \sum_\pi \frac{p\left(u_G | \pi\right)^\infty}{p\left(u_G | \pi^*\right)^\infty} p_1\left(\pi\right). \tag{20}$$

The objective is then to show that the above equals $p(\pi^*)$. It proceeds directly from the definition of $\pi^*$ that, for all $\pi$ other than $\pi^*$,

$$\frac{p\left(u_G | \pi\right)}{p\left(u_G | \pi^*\right)} < 1 \tag{21}$$

Thus, all but one of the terms in the sum above approach zero, and the remaining term equals $p_1(\pi^*)$. Thus,

$$\sum_\pi \frac{p\left(u_G | \pi\right)^\infty}{p\left(u_G | \pi^*\right)^\infty} p_1\left(\pi\right) = p_1\left(\pi^*\right) \quad \square \tag{22}$$

# 3  Simulations

## 3.1  Binary choice

We begin with a simulation of a simple incentive choice situation. Here, an animal faces two levers. Pressing the left lever reliably yields a preferred food ($r = 2$), the right a less preferred food ($r = 1$). Representing these contingencies in a network structured as in Fig. 1 (left) and employing the iterative algorithm described in section 2.2 yields the results in Figure 2A. Shown here are the posterior probabilities for the policies *press left* and *press right*, along with the marginal value of $p(u = 1)$ under these posteriors (labeled *EV* for expected value). The dashed horizontal line indicates the expected value for the optimal plan, to which the model obviously converges.

A key empirical assay for purposive behavior involves *outcome devaluation*. Here, actions yielding a previously valued outcome are abandoned after the incentive value of the outcome is reduced, for example by pairing with an aversive event (e.g., [4]). To simulate this within the binary choice scenario just described, we reduced to zero the reward value of the food yielded by the left lever ($f_L$), by making the appropriate change to $p(u|f_L)$. This yielded a reversal in lever choice (Fig. 2B).

Another signature of purposive actions is that they are abandoned when their causal connection with rewarding outcomes is removed (*contingency degradation,* see [4]). We simulated this by starting with the model from Fig. 2A and changing conditional probabilities at $s$ for $t=2$ to reflect a decoupling of the *left* action from the $f_L$ outcome. The resulting behavior is shown in Fig. 2C.

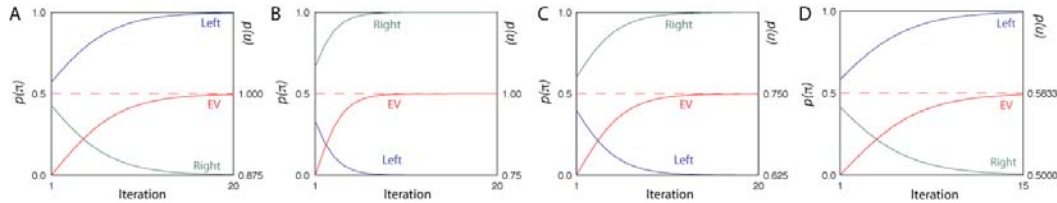

Fig 2. Simulation results, binary choice.

## 3.2  Stochastic outcomes

A critical aspect of the present modeling paradigm is that it yields reward-maximizing choices in stochastic domains, a property that distinguishes it from some other recent approaches using graphical models to do planning (e.g., [9]). To illustrate, we used the architecture in Figure 1 (left) to simulate a choice between two fair coins. A 'left' coin yields $1 for heads, $0 for tails; a 'right' coin $2 for heads but for tails a $3 *loss*. As illustrated in Fig. 2D, the model maximizes expected value by opting for the left coin.

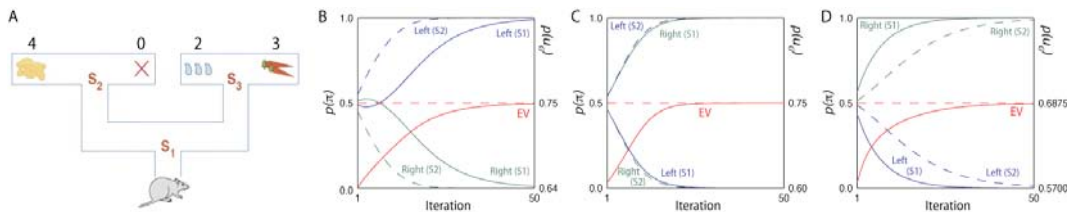

Fig 3. Simulation results, two-step sequential choice.

## 3.3  Sequential decision

Here, we adopt the two-step T-maze scenario used by [24] (Fig. 3A). Representing the task contingencies in a graphical model based on the template from Fig 1 (right), and using the reward values indicated in Fig. 3A, yields the choice behavior shown in Figure 3B. Following [24], a shift in motivational state from hunger to thirst can be represented in the

graphical model by changing the reward function ($R$(cheese) = 2, $R$(X) = 0, $R$(water) = 4, $R$(carrots) = 1). Imposing this change at the level of the $u$ variables yields the choice behavior shown in Fig. 3C. The model can also be used to simulate effort-based decision. Starting with the scenario in Fig. 2A, we simulated the insertion of an effort-demanding scalable barrier at $S_2$ ($R(S_2)$ = -2) by making appropriate changes $p(u|s)$. The resulting behavior is shown in Fig. 3D.

A famous empirical demonstration of purposive control involves *detour behavior*. Using a maze like the one shown in Fig. 4A, with a food reward placed at $s_5$, Tolman [2] found that rats reacted to a barrier at location $A$ by taking the upper route, but to a barrier at $B$ by taking the longer lower route. We simulated this experiment by representing the corresponding transition and reward functions in a graphical model of the form shown in Fig. 1 (right),[3] representing the insertion of barriers by appropriate changes to the transition function. The resulting choice behavior at the critical juncture $s_2$ is shown in Fig. 4.

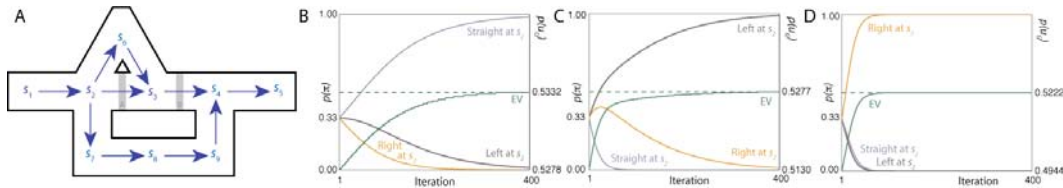

Fig 4. Simulation results, detour behavior. B: No barrier. C: Barrier at $A$. D: Barrier at $B$.

Another classic empirical demonstration involves *latent learning*. Blodgett [25] allowed rats to explore the maze shown in Fig. 5. Later insertion of a food reward at $s_{13}$ was followed immediately by dramatic reductions in the running time, reflecting a reduction in entries into blind alleys. We simulated this effect in a model based on the template in Fig. 1 (right), representing the maze layout via an appropriate transition function. In the absence of a reward at $s_{12}$, random choices occurred at each intersection. However, setting $R(s_{13})$ = 1 resulted in the set of choices indicated by the heavier arrows in Fig. 5.

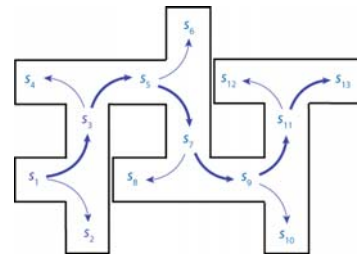

Fig 5. Latent learning.

## 4    Relation to previous work

Initial proposals for how to solve decision problems through probabilistic inference in graphical models, including the idea of encoding reward as the posterior probability of a random utility variable, were put forth by Cooper [23]. Related ideas were presented by Shachter and Peot [12], including the use of nodes that integrate information from multiple utility nodes. More recently, Attias [11] and Verma and Rao [9] have used graphical models to solve shortest-path problems, leveraging probabilistic representations of rewards, though not in a way that guaranteed convergence on optimal (reward maximizing) plans. More closely related to the present research is work by Toussaint and Storkey [10], employing the EM algorithm. The iterative approach we have introduced here has a certain resemblance to the EM procedure, which becomes evident if one views the policy variables in our models as parameters on the mapping from states to actions. It seems possible that there may be a formal equivalence between the algorithm we have proposed and the one reported by [10].

As a cognitive and neuroscientific proposal, the present work bears a close relation to recent work by Hasselmo [6], addressing the prefrontal computations underlying goal-directed action selection (see also [7]). The present efforts are tied more closely to normative principles of decision-making, whereas the work in [6] is tied more closely to the details of neural circuitry. In this respect, the two approaches may prove complementary, and it will be interesting to further consider their interrelations.

## Acknowledgments

Thanks to Andrew Ledvina, David Blei, Yael Niv, Nathaniel Daw, and Francisco Pereira for useful comments.

## Footnotes

[1] Note that temporal discounting can be incorporated into the framework through minimal modifications to Equation 2.

[2] In the single-action situation, where there is only one *u* node, it is this variable that is treated as observed (*u* = 1).

[3] In this simulation and the next, the set of states associated with each state node was limited to the set of reachable states for the relevant time-step, assuming an initial state of $s_1$.

## References

[1] Hull, C.L., *Principles of Behavior*. 1943, New York: Appleton-Century.

[2] Tolman, E.C., *Purposive Behavior in Animals and Men*. 1932, New York: Century.

[3] Dickinson, A., *Actions and habits: the development of behavioral autonomy*. Philosophical Transactions of the Royal Society (London), Series B, 1985. **308**: p. 67-78.

[4] Balleine, B.W. and A. Dickinson, *Goal-directed instrumental action: contingency and incentive learning and their cortical substrates*. Neuropharmacology, 1998. **37**: p. 407-419.

[5] Daw, N.D., Y. Niv, and P. Dayan, *Uncertainty-based competition between prefrontal and striatal systems for behavioral control*. Nature Neuroscience, 2005. **8**: p. 1704-1711.

[6] Hasselmo, M.E., *A model of prefrontal cortical mechanisms for goal-directed behavior*. Journal of Cognitive Neuroscience, 2005. **17**: p. 1115-1129.

[7] Schmajuk, N.A. and A.D. Thieme, *Purposive behavior and cognitive mapping.  A neural network model*. Biological Cybernetics, 1992. **67**: p. 165-174.

[8] Tatman, J.A. and R.D. Shachter, *Dynamic programming and influence diagrams*. IEEE Transactions on Systems, Man and Cybernetics, 1990. **20**: p. 365-379.

[9] Verma, D. and R.P.N. Rao. *Planning and acting in uncertain enviroments using probabilistic inference*. in *IEEE/RSJ International Conference on Intelligent Robots and Systems*. 2006.

[10] Toussaint, M. and A. Storkey. *Probabilistic inference for solving discrete and continuous state markov decision processes*. in *Proceedings of the 23rd International Conference on Machine Learning*. 2006. Pittsburgh, PA.

[11] Attias, H. *Planning by probabilistic inference*. in *Proceedings of the 9th Int. Workshop on Artificial Intelligence and Statistics*. 2003.

[12] Shachter, R.D. and M.A. Peot. *Decision making using probabilistic inference methods*. in *Uncertainty in artificial intelligence: Proceedings of the Eighth Conference (1992)*. 1992. Stanford University: M. Kaufmann.

[13] Chater, N., J.B. Tenenbaum, and A. Yuille, *Probabilistic models of cognition: conceptual foundations*. Trends in Cognitive Sciences, 2006. **10**(7): p. 287-291.

[14] Doya, K., et al., eds. *The Bayesian Brain: Probabilistic Approaches to Neural Coding*. 2006, MIT Press: Cambridge, MA.

[15] Miller, E.K. and J.D. Cohen, *An integrative theory of prefrontal cortex function*. Annual Review of Neuroscience, 2001. **24**: p. 167-202.

[16] Asaad, W.F., G. Rainer, and E.K. Miller, *Task-specific neural activity in the primate prefrontal cortex*. Journal of  Neurophysiology, 2000. **84**: p. 451-459.

[17] Rolls, E.T., *The functions of the orbitofrontal cortex*. Brain and Cognition, 2004. **55**: p. 11-29.

[18] Padoa-Schioppa, C. and J.A. Assad, *Neurons in the orbitofrontal cortex encode economic value*. Nature, 2006. **441**: p. 223-226.

[19] Gopnik, A., et al., *A theory of causal learning in children: causal maps and Bayes nets*. Psychological Review, 2004. **111**: p. 1-31.

[20] Hamilton, A.F.d.C. and S.T. Grafton, *Action outcomes are represented in human inferior frontoparietal cortex*. Cerebral Cortex, 2008. **18**: p. 1160-1168.

[21] Johnson, A., M.A.A. van der Meer, and D.A. Redish, *Integrating hippocampus and striatum in decision-making*. Current Opinion in Neurobiology, 2008. **17**: p. 692-697.

[22] Jensen, F.V., *Bayesian Networks and Decision Graphs*. 2001, New York: Springer Verlag.

[23] Cooper, G.F. *A method for using belief networks as influence diagrams*. in *Fourth Workshop on Uncertainty in Artificial Intelligence*. 1988. University of Minnesota, Minneapolis.

[24] Niv, Y., D. Joel, and P. Dayan, *A normative perspective on motivation*. Trends in Cognitive Sciences, 2006. **10**: p. 375-381.

[25] Blodgett, H.C., *The effect of the introduction of reward upon the maze performance of rats*. University of California Publications in Psychology, 1929. **4**: p. 113-134.